# Resolving motion ambiguities

**K. I. Diamantaras**
Siemens Corporate Research
755 College Rd. East
Princeton, NJ 08540

**D. Geiger***
Courant Institute, NYU
Mercer Street
New York, NY 10012

## Abstract

We address the problem of optical flow reconstruction and in particular the problem of resolving ambiguities near edges. They occur due to (i) the aperture problem and (ii) the occlusion problem, where pixels on both sides of an intensity edge are assigned the same velocity estimates (and confidence). However, these measurements are correct for just one side of the edge (the non occluded one).

Our approach is to introduce an uncertainty field with respect to the estimates and confidence measures. We note that the confidence measures are large at intensity edges and larger at the convex sides of the edges, i.e. inside corners, than at the concave side. We resolve the ambiguities through local interactions via coupled Markov random fields (MRF). The result is the detection of motion for regions of images with large global convexity.

## 1 Introduction

In this paper we discuss the problem of figure ground separation, via optical flow, for homogeneous images (textured images just provide more information for the disambiguation of figure-ground). We address the problem of optical flow reconstruction and in particular the problem of resolving ambiguities near intensity edges. We concentrate on a two frames problem, where all the motion ambiguities we discuss can be disambiguiated by the human visual system.

Optical flow is a 2D (two dimensional) field defined as to capture the projection of the 3D (three dimensional) motion field into the view plane (retina). The Horn and Schunk[8] formulation of the problem is to impose (i) the brightness constraint $\frac{dE(x,y,t)}{dt} = 0$, where $E$ is the intensity image, and (ii) the smoothness of the velocity field. The smoothness can be thought of coming from a rigidity or quasi-rigidity assumption (see Ullman [12]).

We utilize two improvements which are important for the optical flow computation, (i) the introduction of the confidence measure (Nagel and Enkelman [10], Anandan [1]) and (ii) the application of smoothness while preserving discontinuities (Geman and Geman [6], Blake and Zisserman [2], Mumford and Shah [9]). It is clear that as an object moves with respect to a background not only optical flow discontinuities occur, but also occlusions occur (and revelations). In stereo, occlusions are related to discontinuities (e.g. Geiger et. al 1992 [5]), and for motion a similar relation must exist. We study ambiguities ocuring at motion discontinuities and occlusions in images.

The paper is organized as follows: Section 2 describes the problem with examples and a brief discussion on possible approaches, section 3 presents our approach, with the formulation of the model and a method to solve it, section 4 gives the results.

## 2    Motion ambiguities

Figure 1 shows two synthetic problems involving a translation and a rotation of simple objects in front of stationary backgrounds.

Consider the case of the square translation (see figure 1a.). Humans perceive the square translating, although block-matching (and any other matching technique) gives translation on both sides of the square edges. Moreover, there are other interpretations of the scene, such as the square belonging to the stationary background and the outside being a translating foreground with a square hole. The examples are synthetic, but emphasize the ambiguities. Real images may have more texture, thus many times helping resolve these ambiguities, but not everywhere.

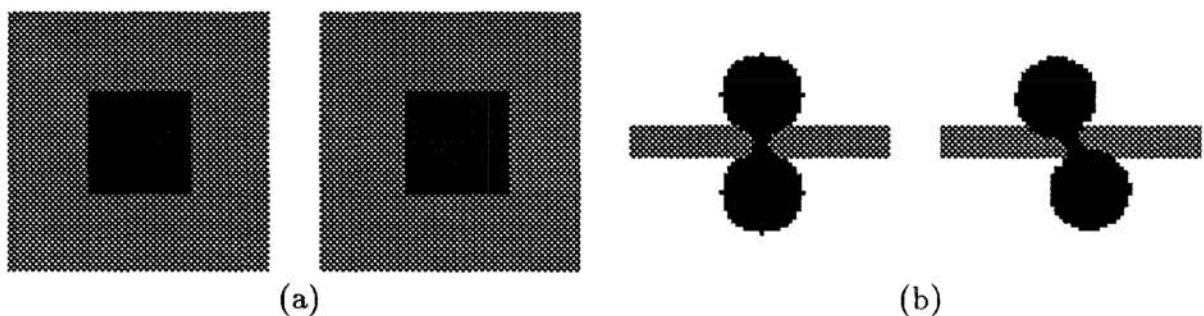

(a)                                                    (b)

Figure 1: Two image sequences of $128 \times 128$. (a) Square translation of 3 pixels; (b) "Eight" rotation of $10°$. Note that the "eight" has concave and convex regions.

# 3    A Markov random field model

We describe a model capable of solving these ambiguities. It is based on coupled Markov random fields and thus, based on local processes. Our main contribution is to introduce the idea of uncertainty on the estimates and confidence measures. We propose a Markov field that allows the estimates of each pixel to be chosen among a large neighborhood, thus each pixel estimate can be neglected. We show that convex regions of the image do bias the confidence measures such that the final motion solutions are expected to be the ones with global larger convexity  Note that locally, one can have concave regions of a shape that give "wrong" bias (see figure 1 b).

## 3.1    Block Matching

Block matching is the process of correlating a block region of one image, say of size $(2\omega_M+1)\times(2\omega_M+1)$, with a block region of the other image. Block-matching yields a set of matching errors $d_{ij}^{mn}$, where $(i,j)$ is a pixel in the image and $v = [m,n]$ is a displacement vector in a search window of size $(2\omega_S+1)\times(2\omega_S+1)$ around the pixel. We define the velocity measurements $g_{ij}$ and the covariance matrix $C_{ij}$ as the mean and variance of the vector $v = [m,n]$ averaged according to the distribution $e^{-kd_{ij}^{mn}}$ :

$$g_{ij} = \frac{\sum_{m,n} e^{-kd_{ij}^{mn}} v}{\sum_{m,n} e^{-kd_{ij}^{mn}}} \qquad C_{ij} = \frac{\sum_{m,n} e^{-kd_{ij}^{mn}} (v-g_{ij})(v-g_{ij})^T}{\sum_{m,n} e^{-kd_{ij}^{mn}}}$$

Figure 2 shows the block matching data $g_{ij}$ for the two problems discussed above and figure 3 shows the correspondent confidence measurse (inverse of the covariance matrix as defined below).

## 3.2    The aperture problem and confidence

The aperture problem [7] occurs where there is a low confidence on the measurements (data) in the direction along an edge; In particular we follow the approach by [1].

The eigenvalues $\lambda_1, \lambda_2$, of $C_{ij}$ correspond to the variance of distribution of $v$ along the directions of the corresponding eigenvectors $v_1, v_2$. The confidence of the estimate should be inversely proportional to the variance of the distribution, i.e. the confidence along direction $v_1$ ($v_2$) is $\propto 1/\lambda_1$ ($\propto 1/\lambda_2$). All this confidence information can be packaged inside the *confidence matrix* defined as follows:

$$R_{ij} = \epsilon(C_{ij} + \epsilon)^{-1} \tag{1}$$

where $\epsilon$ is a very small constant that guarantees invertibility. Thus the eigenvalues of $R_{ij}$ are values between 0 and 1 corresponding to the confidence along the directions $v_1$ and $v_2$, whereas $v_1$ and $v_2$ are still eigenvectors of $R_{ij}$.

The confidence measures at straight edges is high perpendincular to the edges and low (zero) along the edges. However, at corners, the confidence is high on both

directions thus through smoothness this result can be propagated through the other parts of the image, then resolving the aperture problem.

### 3.3    The localization problem and a binary decision field

The localization problem arises due to the local symmetry at intensity edges, where both sides of an edge give the same correspondences. These cases occur when occluded regions are homogeneous and so, block matching, pixel matching or any matching technique can not distinguish which side of the edge is being occluded or is occluding. Even if one considers edge based methods, the same problem arises in the reconstruction stage, where the edge velocities have to be propagated to the rest of the image. In this cases a localization uncertainty is introduced. More precisely, pixels whose matching block contains a strong feature (e.g. a corner) will obtain a high-confidence motion estimate along the direction in which this feature moved. Pixels on both sides of this feature, and at distances less than half the matching window size, $\omega_M$, will receive roughly the same motion estimates associated with high confidences. However, it could have been just one of the two sides that have moved in this direction. In that case this estimate should not be taken into account on the other side. We note however a bias towards inside of corner regions from the confidence measures.

Note that in a corner, despite both sides getting roughly the same velocity estimate and high confidence measures, the inside pixel always get a larger confidence. This bias is due to having more pixels outside the edge of a closed contour than outside, and occurs at the convex regions (e.g. a corner). Thus, in general, the convex regions will have a stronger confidence measure than outside them. Note that at concavities in the "eight" rotation image, the confidence will be higher outside the "eight" and correct at convex regions. Thus, a global optimization will be required to decide which confidences to "pick up".

Our approach to resolve this ambiguity is to allow for the motion estimate at pixel $(i, j)$ to select data from a neighborhood $N_{ij}$, and its goal is to maximize the total estimates (taking into account the confidence measures). More precisely, let $f_{ij}$ be the vector motion field at pixel $(i, j)$. We introduce a binary field $\alpha_{ij}^{mn}$ that indicates which data $g_{i+m, j+n}$ in a neighborhood $N_{ij}$ of $(i, j)$ should correspond to a motion estimate $f_{ij}$. The size of $N_{ij}$ is given by $\omega_M + 1$ to overcome the localization uncertainty. For a given lattice point $(i, j)$ the boolean parameters $\alpha_{ij}^{mn}$ should be mutually exclusive, i.e. only one of them, $\alpha_{ij}^{m^*n^*}$, should be equal to 1 indicating that $f_{ij}$ should correspond to $g_{i+m^*, j+n^*}$, while the rest $\alpha_{ij}^{mn}$, $m \neq m^*$, $n \neq n^*$, should be zero (or $\sum_{m^*n^* \in N_{ij}} \alpha_{ij}^{m^*n^*} = 1$). The conditional probability reflects both an uncertainty due to noise and an uncertainty due to spatial localization of the data

$$P(R, g | f, \alpha) = \frac{1}{C_2} \exp\{-\sum_{ij} \sum_{mn \in N_{ij}} \alpha_{ij}^{mn} \| R_{i+m, j+n}(f_{ij} - g_{i+m, j+n}) \|^2\} \quad (2)$$

where $\|h\|^2 = h_x^2 + h_y^2$ for $h = [h_x, h_y]$.

### 3.4 The piecewise smooth prior

The prior probability of the motion field $f_{ij}$ is a piecewise smoothness condition, as in [6].

$$P(f, \alpha, h, v) = \frac{1}{C_1} \exp\left\{ -\left( \sum_{ij} \mu(1 - h_{ij}) \| f_{ij} - f_{i-1,j} \|^2 + \mu(1 - v_{ij}) \| f_{ij} - f_{i,j-1} \|^2 + \gamma_{ij}(h_{ij} + v_{ij}) \right) \right\}.$$

(3)

where $h_{ij} = 0$ ($v_{ij} = 0$) if there is no motion discontinuity separating pixels $(i, j)$, $(i - 1, j)$ $((i, j), (i, j - 1))$ , otherwise $h_{ij} = 1$ ($v_{ij} = 1$). The parameter $\mu$ has to be estimated. We have considered that the cost to create motion discontinuities should be lowered at intensity edges (see Poggio et al. [11]), i.e $\gamma_{ij} = \gamma(1 - \delta e_{ij})$, where $e_{ij}$ is the intensity edge and $0 \leq \delta \leq 1$ and $\gamma$ have to be estimated.

### 3.5 The posterior distribution

The posterior distribution is given by Bayes' law

$$P(f, \alpha, h, v | g, R) = \frac{1}{P(g, R)} P(g, R | f, \alpha) P(f, \alpha, h, v) = \frac{1}{Z} e^{-V(f,\alpha,h,v;g)} \quad (4)$$

where

$$
\begin{aligned}
V(f, \alpha, h, v) &= \sum_{ij} \left\{ \sum_{mn \in N_{ij}} \alpha_{ij}^{mn} \| R_{i+m,j+n}(f_{ij} - g_{i+m,j+n}) \|^2 \right. \\
&+ \mu(1 - h_{ij}) \| f_{ij} - f_{i-1,j} \|^2 + \mu(1 - v_{ij}) \| f_{ij} - f_{i,j-1} \|^2 + \\
&\left. \gamma_{ij}(h_{ij} + v_{ij}) \right\}
\end{aligned}
$$

(5)

is the energy of the system. Ideally, we would like to minimize $V$ under all possible configurations of the fields $f$, $h$, $v$ and $\alpha$, while obeying the constraint $\sum_{mn \in N_{ij}} \alpha_{ij}^{mn} = 1$.

### 3.6 Mean field techniques

Introducing the inverse temperature parameter $\beta (= 1/T)$ we can obtain the transformed probability distribution

$$P_\beta(f, \alpha | g, R) = \frac{1}{Z_\beta} e^{-\beta V(f,\alpha)} \quad (6)$$

where

$$
\begin{aligned}
Z_\beta &= \sum_{\{f\}} \exp\{ -\beta \sum_{ij} \mu_{ij}^h \| f_{ij} - f_{i-1,j} \|^2 + \mu_{ij}^v \| f_{ij} - f_{i,j-1} \|^2 \} \\
&\times \left( \sum_{\{\alpha\}} \exp\{ -\beta \sum_{ij} \sum_{mn \in N_{ij}} \alpha_{ij}^{mn} \| R_{i+m,j+n}(f_{ij} - g_{i+m,j+n}) \|^2 \} \right)
\end{aligned}
$$

(7)

where $\mu_{ij}^v = \mu(1 - \bar{v}_{ij})$ and $\mu_{ij}^h = \mu(1 - \bar{h}_{ij})$.

We have to obey the constraint $\sum_{mn \in N_{ij}} \alpha_{ij}^{mn} = 1$. For the sake of simplicity we have assumed that the neighborhood $N_{ij}$ around site $(i, j)$ is $N_{ij} = \{(i + m, j + n) : -1 \leq m \leq 1, -1 \leq n \leq 1\}$. The second factor in (7) can be explicitly computed. Employing the mean field techniques proposed in [3] and extended in [4] we can average out the variables $h$, $v$ and $\alpha$ (including the constraint) and yield

$$Z_\beta = \sum_{\{f\}} \prod_{ij} \Big( \sum_{m,n=-1}^{1} (\exp\{-\beta\|R_{i+m,j+n}(f_{ij} - g_{i+m,j+n})\|^2\})$$
$$(e^{\gamma_{ij}} + e^{\mu\|f_{ij} - f_{i-1,j}\|^2})(e^{\gamma_{ij}} + e^{\mu\|f_{ij} - f_{i,j-1}\|^2}) \Big) \tag{8}$$

which yields the following effective energy

$$V_{eff}(f) = -\frac{1}{\beta} \sum_{ij} \ln\Big( \sum_{m,n=-1}^{1} \exp\{-\beta\|R_{i+m,j+n}(f_{ij} - g_{i+m,j+n})\|^2\}\Big) +$$
$$\ln(e^{\gamma_{ij}} + e^{\mu\|f_{ij}-f_{i-1,j}\|^2})(e^{\gamma_{ij}} + e^{\mu\|f_{ij}-f_{i,j-1}\|^2})$$

since $Z_\beta = \sum_{\{f\}} e^{-\beta V_{eff}(f)}$. Using the saddle point approximation, i.e. considering $Z_\beta \approx e^{-\beta V_{eff}(\bar{f})}$ with $\bar{f}$ minimizing $V_{eff}(f; g)$, the mean field equations become

$$0 = \sum_{mn} \bar{\alpha}_{ij}^{mn} R_{i+m,j+n}(\bar{f}_{ij} - g_{i+m,j+n}) + \mu_{ij}^v \Delta^v f_{ij} + \mu_{ij}^h \Delta^h f_{ij}$$

with $\mu_{ij}^v = \mu(1 - \bar{v}_{ij})$ and $\mu_{ij}^h = \mu(1 - \bar{h}_{ij})$, $\Delta^h f_{ij} = (\bar{f}_{ij} - \bar{f}_{i-1,j})$, $\Delta^v f_{ij} = (\bar{f}_{ij} - \bar{f}_{i,j-1})$, and

$$\bar{\alpha}_{ij}^{mn} = \frac{e^{-\beta\|R_{i+m,j+n}(\bar{f}_{ij} - g_{i+m,j+n})\|^2}}{\sum_{m^*n^* \in N_{ij}} e^{-\beta\|R_{i+m^*,j+n^*}(\bar{f}_{ij} - g_{i+m^*,j+n^*})\|^2}},$$
$$\bar{h}_{ij} = \frac{1}{1 + e^{\gamma_{ij} - \mu\|\Delta^h f_{ij}\|^2}} \quad, \text{and} \quad \bar{v}_{ij} = \frac{1}{1 + e^{\gamma_{ij} - \mu\|\Delta^v f_{ij}\|^2}} \tag{9}$$

The normalization constant $Z_\beta$ called the partition function, has the important property that

$$\lim_{\beta \to \infty} -\frac{1}{\beta} \ln Z_\beta = \min_{\{f,\alpha,h,v\}} \{V(f, \alpha, h, v)\} \tag{10}$$

Then using an annealing method we let $\beta \to \infty$ and the minimum of $V_\beta = -\frac{1}{\beta} \ln Z_\beta$ approaches asymptotically the desired minimum.

## 4  Results

We have applied an iterative method along with an annealing schedule to solve the above mean field equations for $\beta \to \infty$. The method was run on the two examples already described. Figure 4 depicts the results of the experiments. The system chooses a natural interpretation (in agreement with human perception), namely it interprets the object (e.g. the square in the first example or the eight-shaped region in the second example) moving and the background being stationary. In the beginning of the annealing process the localization field $\alpha$ may produce "erroneous" results, however the neighbor information eventually forces the pixels outside the moving object to coincide with the rest of the background which has zero motion. For the pixels inside the object, on the contrary, the neighbor information eventually reinforces the adoption of the motion of the edges.

## Footnotes

*work done when the author was at the Isaac Newton Institute and at Siemens Corporate Research

## References

[1]  P. Anandan, "Measuring Visual Motion from Image Sequences", PhD thesis. COINS Dept., Univ. Massachusetts, Amherst, 1987.

[2]  A. Blake and A. Zisserman, "Visual Reconstruction", Cambridge, Mass, MIT press, 1987.

[3]  D. Geiger and F. Girosi, "Parallel and Deterministic Algorithms for MRFs: Surface Reconstruction and Integration", IEEE PAMI: 13(5), May 1991.

[4]  D. Geiger and A. Yuille, "A Common Framework for Image Segmentation", Int. J. Comput. Vision, 6(3), pp. 227–243, 1991.

[5]  D. Geiger and B. Ladendorf and A. Yuille, "Binocular stereo with occlusion", Computer Vision- ECCV92, ed. G. Sandini, Springer-Verlag, 588, pp 423–433, May 1992.

[6]  S. Geman and D. Geman, "Stochastic Relaxation, Gibbs Distributions, and the Bayesian Restoration of Images", IEEE PAMI 6, pp. 721–741, 1984.

[7]  E. C. Hildreth, "The measurement of visual motion", MIT press, 1983.

[8]  B.K.P. Horn and B.G. Schunk, "Determining optical flow", Artificial Intelligence, vol 17, pp. 185–203, August 1981.

[9]  D. Mumford and J. Shah, "Boundary detection by minimizing functionals, I", Proc. IEEE Conf. on Computer Vision & Pattern Recognition, San Francisco, CA, 1985.

[10]  H.-H. Nagel and W. Enkelmann, "An Investigation of Smoothness Constraints for the Estimation of Displacement Vector Fields from Image Sequences", IEEE PAMI: 8, 1986.

[11]  T. Poggio and E. B. Gamble and J. J. Little, "Parallel Integration of Vision Module", Science, vol 242, pp. 436–440, 1988.

[12]  S. Ullman, "The Interpretation of Visual Motion", Cambridge, Mass, MIT press, 1979.

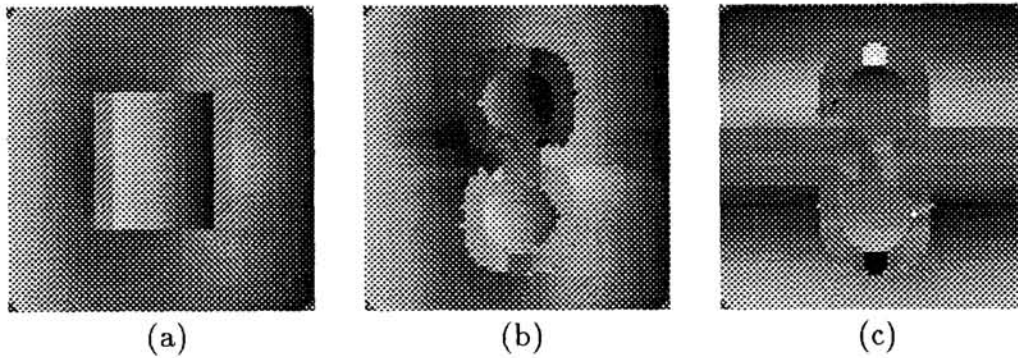

Figure 2: Block matching data $g_{ij}$. Both sides of the edges have the same data (and same confidence). White represents motion to the right (x-direction) or up (y-direction). Black is the complement. (a) The x-component of the data for the square translation. (b) The x-component of the data for the rotation and (c) the y-component of the data.

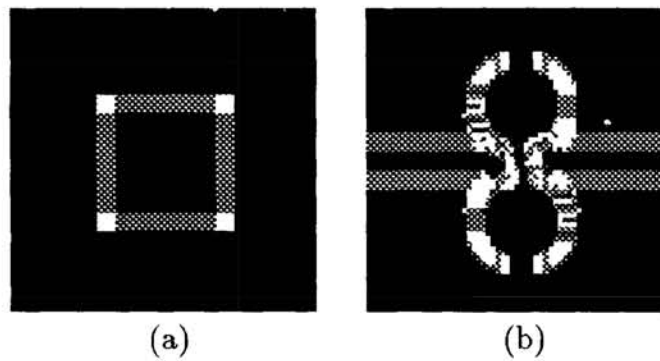

Figure 3: The confidence $R$ extracted from the block matching data $g_{ij}$. The display is the sum of both eigenvalues, i.e. the trace of R. Both sides of the edges have the same confidence. White represents high confidence. (a) For the square translation. (b) For the rotation.

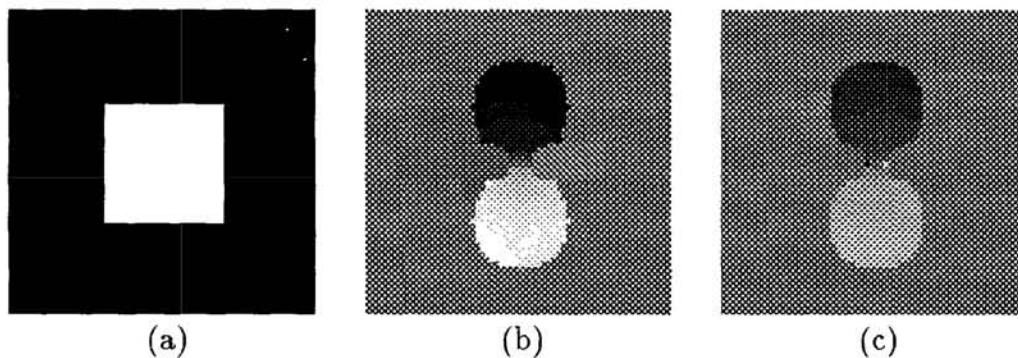

Figure 4: The final motion estimation, after 20000 iterations, resolved the ambiguities with a natural interpretation of the scene. $\mu = 10$, $\delta = 1$, $\gamma = 100$. (a) square translation (b) x component of the motion rotation (c) y component of the motion rotation